# A computational model of hippocampal function in trace conditioning

**Elliot A. Ludvig, Richard S. Sutton, Eric Verbeek**
Department of Computing Science
University of Alberta
Edmonton, AB, Canada T6G 2E8
{elliot,sutton,everbeek}@cs.ualberta.ca

**E. James Kehoe**
School of Psychology
University of New South Wales
Sydney, NSW, Australia 2052
j.kehoe@unsw.edu.au

## Abstract

We introduce a new reinforcement-learning model for the role of the hippocampus in classical conditioning, focusing on the differences between trace and delay conditioning. In the model, all stimuli are represented both as unindividuated wholes and as a series of temporal elements with varying delays. These two stimulus representations interact, producing different patterns of learning in trace and delay conditioning. The model proposes that hippocampal lesions eliminate long-latency temporal elements, but preserve short-latency temporal elements. For trace conditioning, with no contiguity between cue and reward, these long-latency temporal elements are necessary for learning adaptively timed responses. For delay conditioning, the continued presence of the cue supports conditioned responding, and the short-latency elements suppress responding early in the cue. In accord with the empirical data, simulated hippocampal damage impairs trace conditioning, but not delay conditioning, at medium-length intervals. With longer intervals, learning is impaired in both procedures, and, with shorter intervals, in neither. In addition, the model makes novel predictions about the response topography with extended cues or post-training lesions. These results demonstrate how temporal contiguity, as in delay conditioning, changes the timing problem faced by animals, rendering it both easier and less susceptible to disruption by hippocampal lesions.

The hippocampus is an important structure in many types of learning and memory, with prominent involvement in spatial navigation, episodic and working memories, stimulus configuration, and contextual conditioning. One empirical phenomenon that has eluded many theories of the hippocampus is the dependence of aversive trace conditioning on an intact hippocampus (but see Rodriguez & Levy, 2001; Schmajuk & DiCarlo, 1992; Yamazaki & Tanaka, 2005). For example, trace eyeblink conditioning disappears following hippocampal lesions (Solomon et al., 1986; Moyer, Jr. et al., 1990), induces hippocampal neurogenesis (Gould et al., 1999), and produces unique activity patterns in hippocampal neurons (McEchron & Disterhoft, 1997). In this paper, we present a new abstract computational model of hippocampal function during trace conditioning. We build on a recent extension of the temporal-difference (TD) model of conditioning (Ludvig, Sutton & Kehoe, 2008; Sutton & Barto, 1990) to demonstrate how the details of stimulus representation can qualitatively alter learning during trace and delay conditioning. By gently tweaking this stimulus representation and reducing long-latency temporal elements, trace conditioning is severely impaired, whereas delay conditioning is hardly affected. In the model, the hippocampus is responsible for maintaining these long-latency elements, thus explaining the selective importance of this brain structure in trace conditioning.

The difference between trace and delay conditioning is one of the most basic operational distinctions in classical conditioning (e.g., Pavlov, 1927). Figure 1 is a schematic of the two training procedures. In trace conditioning, a conditioned stimulus (CS) is followed some time later by a reward or uncon-

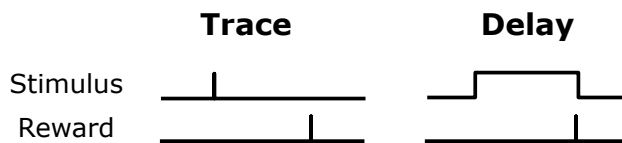

Figure 1: Event timelines in trace and delay conditioning. Time flows from left-to-right in the diagram. A vertical bar represents a punctate (short) event, and the extended box is a continuously available stimulus. In delay conditioning, the stimulus and reward overlap, whereas, in trace conditioning, there is a stimulus-free gap between the two punctate events.

ditioned stimulus (US); the two stimuli are separated by a stimulus-free gap. In contrast, in delay conditioning, the CS remains on until presentation of the US. Trace conditioning is learned more slowly than delay conditioning, with poorer performance often observed even at asymptote.

In both eyeblink conditioning (Moyer, Jr. et al., 1990; Solomon et al., 1986; Tseng et al., 2004) and fear conditioning (e.g., McEchron et al., 1998), hippocampal damage severely impairs the acquisition of conditioned responding during trace conditioning, but not delay conditioning. These selective hippocampal deficits with trace conditioning are modulated by the inter-stimulus interval (ISI) between CS onset and US onset. With very short ISIs ($\sim$300 ms in eyeblink conditioning in rabbits), there is little deficit in the acquisition of responding during trace conditioning (Moyer, Jr. et al., 1990). Furthermore, with very long ISIs ($>$1000 ms), delay conditioning is also impaired by hippocampal lesions (Beylin et al., 2001). These interactions between ISI and the hippocampal-dependency of conditioning are the primary data that motivate the new model.

# 1 TD Model of Conditioning

Our full model of conditioning consists of three separate modules: the stimulus representation, learning algorithm, and response rule. The explanation of hippocampal function relies mostly on the details of the stimulus representation. To illustrate the implications of these representational issues, we have chosen the temporal-difference (TD) learning algorithm from reinforcement learning (Sutton & Barto, 1990, 1998) that has become the sine qua non for modeling reward learning in dopamine neurons (e.g., Ludvig et al., 2008; Schultz, Dayan, & Montague, 1997), and a simple, leaky-integrator response rule described below. We use these for simplicity and consistency with prior work; other learning algorithms and response rules might also yield similar conclusions.

## 1.1 Stimulus Representation

In the model, stimuli are not coherent wholes, but are represented as a series of elements or internal *microstimuli*. There are two types of elements in the stimulus representation: the first is the *presence* microstimulus, which is exactly equivalent to the external stimulus (Sutton & Barto, 1990). This microstimulus is available whenever the corresponding stimulus is on (see Fig. 3). The second type of elements are the *temporal microstimuli* or spectral traces, which are a series of successively later and gradually broadening elements (see Grossberg & Schmajuk, 1989; Machado, 1997; Ludvig et al., 2008). Below, we show how the interaction between these two types of representational elements produces different styles of learning in delay and trace conditioning, resulting in differential sensitivity of these procedures to hippocampal manipulation.

The temporal microstimuli are created in the model through coarse coding of a decaying memory trace triggered by stimulus onset. Figure 2 illustrates how this memory trace (left panel) is encoded by a series of basis functions evenly spaced across the height of the trace (middle panel). Each basis function effectively acts as a receptive field for trace height: As the memory trace fades, different basis functions become more or less active, each with a particular temporal profile (right panel). These activity profiles for the temporal microstimuli are then used to generate predictions of the US. For the basis functions, we chose simple Gaussians:

$$f(y, \mu, \sigma) = \frac{1}{\sqrt{2\pi}} exp(-\frac{(y-\mu)^2}{2\sigma^2}).\tag{1}$$

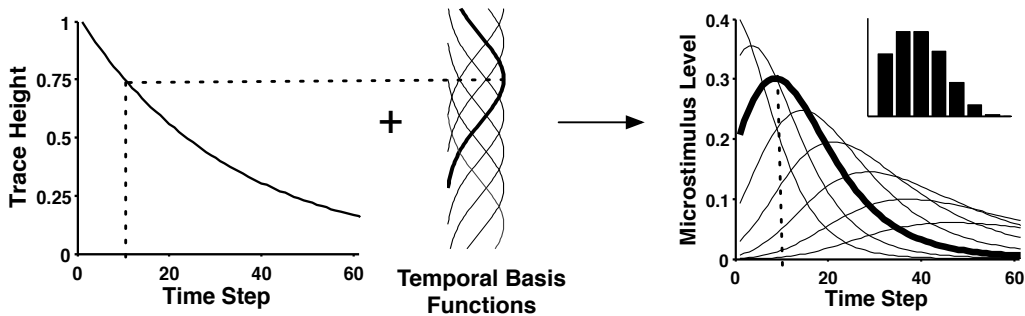

Figure 2: Creating Microstimuli. The memory traces for a stimulus (left) are coarsely coded by a series of temporal basis functions (middle). The resultant time courses (right) of the temporal microstimuli are used to predict future occurrence of the US. A single basis function (middle) and approximately corresponding microstimulus (right) have been darkened. The inset in the right panel shows the levels of several microstimuli at the time indicated by the dashed line.

Given these basis functions, the microstimulus levels $x_t(i)$ at time $t$ are determined by the corresponding memory trace height:

$$x_t(i) = f(y_t, i/m, \sigma)y_t,  \tag{2}$$

where $f$ is the basis function defined above and $m$ is the number of temporal microstimuli per stimulus. The trace level $y_t$ was set to 1 at stimulus onset and decreased exponentially, controlled by a single decay parameter, which was allowed to vary to simulate the effects of hippocampal lesions. Every stimulus, including the US, was represented by a single memory trace and resultant microstimuli.

## 1.2 Hippocampal Damage

We propose that hippocampal damage results in the selective loss of the long-latency temporal elements of the stimulus representation. This idea is implemented in the model through a decrease in the memory decay constant from .985 to .97, approximately doubling the decay rate of the memory trace that determines the microstimuli. In effect, we assume that hippocampal damage results in a memory trace that decays more quickly, or, equivalently, is more susceptible to interference. Figure 3 shows the effects of this parameter manipulation on the time course of the elements in the stimulus representation. The presence microstimulus is not affected by this manipulation, but the temporal microstimuli are compressed for both the CS and the US. Each microstimulus has a briefer time course, and, as a group, they cover a shorter time span. Other means for eliminating or reducing the long-latency temporal microstimuli are certainly possible and would likely be compatible with our theory. For example, if one assumes that the stimulus representation contains multiple memory traces with different time constants, each with a separate set of microstimuli, then eliminating the slower memory traces would also remove the long-latency elements, and many of the results below hold (simulations not shown). The key point is that hippocampal damage reduces the number and magnitude of long-latency microstimuli.

## 1.3 Learning and Responding

The model approaches conditioning as a reinforcement-learning prediction problem, wherein the agent tries to predict the upcoming rewards or USs. The model learns through linear TD($\lambda$) (Ludvig et al., 2008; Schultz et al., 1997; Sutton, 1988; Sutton & Barto, 1990, 1998). At each time step, the US prediction ($V_t$) is determined by:

$$V_t(\mathbf{x}) = \lfloor \mathbf{w}_t^T \mathbf{x} \rfloor_0 = \left\lfloor \sum_{i=1}^{n} w_t(i)x(i) \right\rfloor_0,  \tag{3}$$

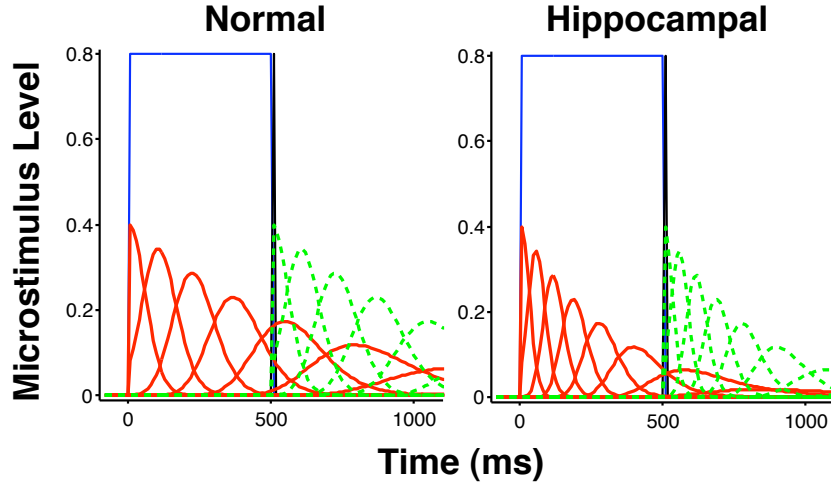

Figure 3: Hippocampal effects on the stimulus representation. The left panel presents the stimulus representation in delay conditioning with the normal parameter settings, and the right panel presents the altered stimulus representation following simulated hippocampal damage. In the hippocampal representation, the temporal microstimuli for both CS (red, solid lines) and US (green, dashed lines) are all briefer and shallower. The presence microstimuli (blue square wave and black spike) are not affected by the hippocampal manipulation.

where $\mathbf{x}$ is a vector of the activation levels $x(i)$ for the various microstimuli, $\mathbf{w}_t$ is a corresponding vector of adjustable weights $w_t(i)$ at time step $t$, and $n$ is the total number of all microstimuli. The US prediction is constrained to be non-negative, with negative values rectified to 0. As is standard in TD models, this US prediction is compared to the reward received and the previous US prediction to generate a TD error ($\delta_t$):

$$\delta_t = r_t + \gamma V_t(\mathbf{x}_t) - V_t(\mathbf{x}_{t-1}), \tag{4}$$

where $\gamma$ is a discount factor that determines the temporal horizon of the US prediction. This TD error is then used to update the weight vector based on the following update rule:

$$\mathbf{w}_{t+1} = \mathbf{w}_t + \alpha \delta_t \mathbf{e}_t, \tag{5}$$

where $\alpha$ is a step-size parameter and $\mathbf{e}_t$ is a vector of eligibility trace levels (see Sutton & Barto, 1998), which together help determine the speed of learning. Each microstimulus has its own corresponding eligibility trace which continuously decays, but accumulates whenever that microstimulus is present:

$$\mathbf{e}_{t+1} = \gamma \lambda \mathbf{e}_t + \mathbf{x}_t, \tag{6}$$

where $\gamma$ is the discount factor as above and $\lambda$ is a decay parameter that determines the plasticity window. These US predictions are translated into responses through a simple, thresholded leaky-integrator response rule:

$$a_{t+1} = \upsilon a_t + \lfloor V_{t+1}(\mathbf{x}_t) \rfloor_\theta, \tag{7}$$

where $\upsilon$ is a decay constant, and $\theta$ is a threshold on the value function $V$.

Our model is defined by Equations 1-7 and 7 additional parameters, which were fixed at the following values for the simulations below: $\lambda = .95$, $\alpha = .005$, $\gamma = .97$, $n = 50$, $\sigma = .08$, $\upsilon = .93$, $\theta = .25$. In the simulated experiments, one time step was interpreted as 10 ms.

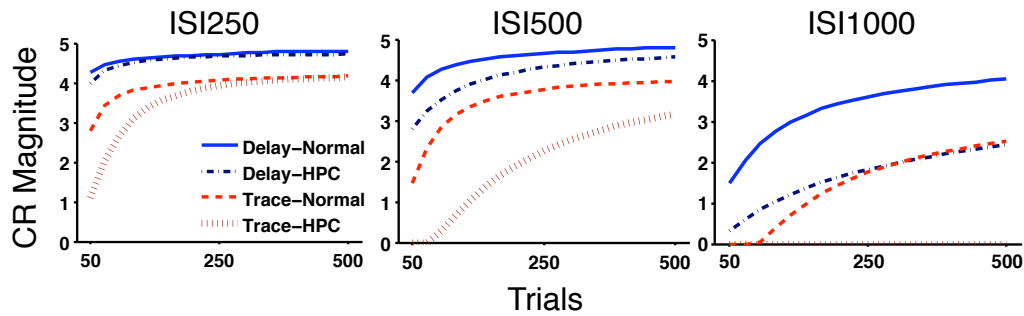

Figure 4: Learning in the model for trace and delay conditioning with and without hippocampal (HPC) damage. The three panels present training with different interstimulus intervals (ISI).

## 2 Results

We simulated 12 total conditions with the model: trace and delay conditioning, both with and without hippocampal damage, for short (250 ms), medium (500 ms), and long (1000 ms) ISIs. Each simulated experiment was run for 500 trials, with every 5th trial an unreinforced probe trial, during which no US was presented. For delay conditioning, the CS lasted the same duration as the ISI and terminated with US presentation. For trace conditioning, the CS was present for 5 time steps (50 ms). The US always lasted for a single time step, and an inter-trial interval of 5000 ms separated all trials (onset to onset). Conditioned responding (CR magnitude) was measured as the maximum height of the response curve on a given trial.

Figure 4 summarizes our results. The figure depicts how the CR magnitude changed across the 500 trials of acquisition training. In general, trace conditioning produced lower levels of responding than delay conditioning, but this effect was most pronounced with the longest ISI. The effects of simulated hippocampal damage varied with the ISI. With the shortest ISI (250 ms; left panel), there was little effect on responding in either trace or delay conditioning. There was a small deficit early in training with trace conditioning, but this difference disappeared quickly with further training. With the longest ISI (1000 ms; right panel), there was a profound effect on responding in both trace and delay conditioning, with trace conditioning completely eliminated. The intermediate ISI (500 ms; middle panel) produced the most complex and interesting results. With this interval, there was only a minor deficit in delay conditioning, but a substantial drop in trace conditioning, especially early in training. This pattern of results roughly matches the empirical data, capturing the selective deficit in trace conditioning caused by hippocampal lesions (Solomon et al., 1986) as well as the modulation of this deficit by ISI (Beylin et al., 2001; Moyer, Jr. et al., 1990).

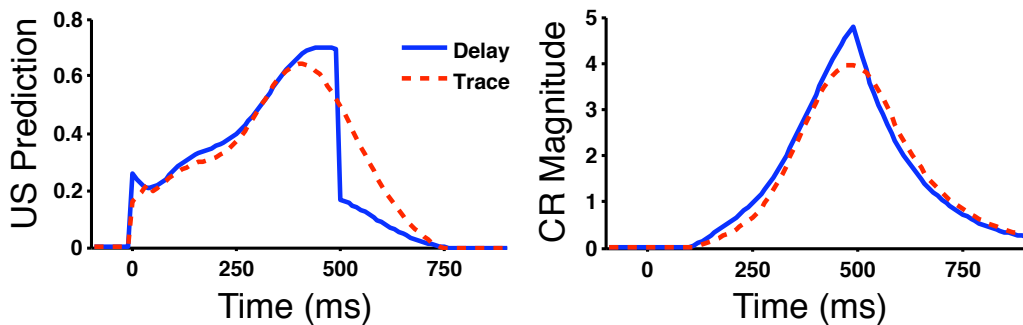

Figure 5: Time course of US prediction and CR magnitude for both trace (red, dashed line) and delay conditioning (blue, solid line) with a 500-ms ISI.

These differences in sensitivity to simulated hippocampal damage arose despite similar model performance during normal trace and delay conditioning. Figure 5 shows the time course of the US prediction (left panel) and CR magnitude (right panel) after trace and delay conditioning on a probe trial with a 500-ms ISI. In both instances, the US prediction grew throughout the trial as the usual time of the US became imminent. Note the sharp drop off in US prediction for delay conditioning exactly as the CS terminates. This change reflects the disappearance of the presence microstimulus, which supports much of the responding in delay conditioning (see Fig. 6). In both procedures, even after the usual time of the US (and CS termination in the case of delay conditioning), there was still some residual US prediction. These US predictions were caused by the long-latency microstimuli, which did not disappear exactly at CS offset, and were ordinarily (on non-probe trials) countered by negative weights on the US microstimuli. The CR magnitude tracked the US prediction curve quite closely, peaking around the time the US would have occurred for both trace and delay conditioning. There was little difference in either curve between trace and delay conditioning, yet altering the stimulus representation (see Fig. 3) had a more pronounced effect on trace conditioning.

An examination of the weight distribution for trace and delay conditioning explains why hippocampal damage had a more pronounced effect on trace than delay conditioning. Figure 6 depicts some representative microstimuli (left column) as well as their corresponding weights (right columns) following trace or delay conditioning with or without simulated hippocampal damage. For clarity in the figure, we have grouped the weights into four categories: positive (+), large positive (+++), negative (-), and large negative (--). The left column also depicts how the model poses the computational problem faced by an animal during conditioning; the goal is to sum together weighted versions of the available microstimuli to produce the ideal US prediction curve in the bottom row. In normal delay conditioning, the model placed a high positive weight on the presence microstimulus, but balanced that with large negative weights on the early CS microstimuli, producing a prediction topography that roughly matched the ideal prediction (see Fig. 5, left panel). In normal trace conditioning, the model only placed a small positive weight on the presence microstimulus, but supplemented that with large positive weights on both the early and late CS microstimuli, also producing a prediction topography that roughly matched the ideal prediction.

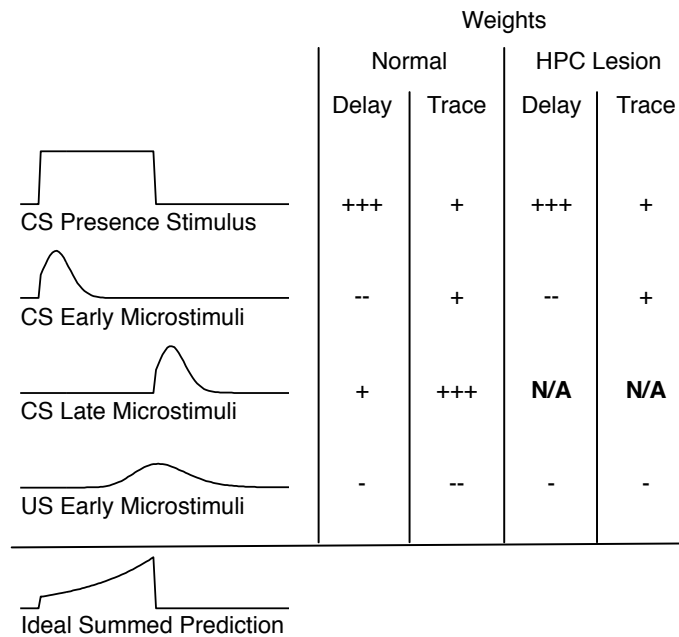

Figure 6: Schematic of the weights (right columns) on various microstimuli following trace and delay conditioning. The left column illustrates four representative microstimuli: the presence microstimulus, an early CS microstimulus, a late CS microstimulus, and a US microstimulus. The ideal prediction is the expectation of the sum of future discounted rewards.

Following hippocampal lesions, the late CS microstimuli were no longer available (N/A), and the system could only use the other microstimuli to generate the best possible prediction profile. In delay conditioning, the loss of these long-latency microstimuli had a small effect, notable only with the longest ISI (1000 ms) with these parameter settings. With trace conditioning, the loss of the long-latency microstimuli was catastrophic, as these microstimuli were usually the major basis for the prediction of the upcoming US. As a result, trace conditioning became much more difficult (or impossible in the case of the 1000-ms ISI), even though delay conditioning was less affected.

The most notable (and defining) difference between trace and delay conditioning is that the CS and US overlap in delay conditioning, but not trace conditioning. In our model, this overlap is necessary, but not sufficient, for the the unique interaction between the presence microstimulus and temporal microstimuli in delay conditioning. For example, if the CS were extended to stay on beyond the time of US occurrence, this contiguity would be maintained, but negative weights on the early CS microstimuli would not suffice to suppress responding throughout this extended CS. In this case, the best solution to predicting the US for the model might be to put high weights on the long-latency temporal microstimuli (as in trace conditioning; see Fig 6), which would not persist as long as the now extended presence microstimulus. Indeed, with a CS that was three times as long as the ISI, we found that the US prediction, CR magnitude, and underlying weights were completely indistinguishable from trace conditioning (simulations not shown). Thus, the model predicts that this extended delay conditioning should be equally sensitive to hippocampal damage as trace conditioning for the same ISIs. This empirical prediction is a fundamental test of the representational assumptions underlying the model.

The particular mechanism that we chose for simulating the loss of the long-latency microstimuli (increasing the decay rate of the memory trace) also leads to a testable model prediction. If one were to pre-train an animal with trace conditioning and then perform hippocampal lesions, there should be some loss of responding, but, more importantly, those CRs that do occur should appear earlier in the interval because the temporal microstimuli now follow a shorter time course (see Fig. 3). There is some evidence for additional short-latency CRs during trace conditioning in lesioned animals (e.g., Port et al., 1986; Solomon et al., 1986), but, to our knowledge, this precise model prediction has not been rigorously evaluated.

## 3 Discussion and Conclusion

We evaluated a novel computational model for the role of the hippocampus in trace conditioning, based on a reinforcement-learning framework. We extended the microstimulus TD model presented by Ludwig et al. (2008) by suggesting a role for the hippocampus in maintaining long-latency elements of the temporal stimulus representation. The current model also introduced an additional element to the stimulus representation (the presence microstimulus) and a simple response rule for translating prediction into actions; we showed how these subtle innovations yield interesting interactions when comparing trace and delay conditioning. In addition, we adduced a pair of testable model predictions about the effects of extended stimuli and post-training lesions.

There are several existing theories for the role of the hippocampus in trace conditioning, including the modulation of timing (Solomon et al., 1986), establishment of contiguity (e.g., Wallenstein et al., 1998), and overcoming of task difficulty (Beylin et al., 2001). Our new model provides a computational mechanism that links these three proposed explanations. In our model, for similar ISIs, delay conditioning requires learning to suppress responding early in the CS, whereas trace conditioning requires learning to create responding later in the trial, near the time of the US (see Fig. 6). As a result, for the same ISI, delay conditioning requires changing weights associated with earlier microstimuli than trace conditioning, though in opposite directions. These early microstimuli reach higher activation levels (see Fig. 2), producing higher eligibility traces, and are therefore learned about more quickly. This differential speed of learning for short-latency temporal microstimuli corresponds with much behavioural data that shorter ISIs tend to improve both the speed and asymptote of learning in eyeblink conditioning (e.g., Schneiderman & Gormezano, 1964). Thus, the contiguity between the CS and US in delay conditioning alters the timing problem that the animal faces, effectively making the time interval to be learned shorter, and rendering the task easier for most ISIs.

In future work, it will be important to characterize the exact mathematical properties that constrain the temporal microstimuli. Our simple Gaussian basis function approach suffices for the datasets

examined here (cf. Ludvig et al., 2008), but other related mathematical functions are certainly possible. For example, replacing the temporal microstimuli in our model with the spectral traces of Grossberg & Schmajuk (1989) produces results that are similar to ours, but using sequences of Gamma-shaped functions tends to fail, with longer intervals learned too slowly relative to shorter intervals. One important characteristic of the microstimulus series seems to be that the heights of individual elements should not decay too quickly. Another key challenge for future modeling is reconciling this abstract account of hippocampal function in trace conditioning with approaches that consider greater physiological detail (e.g., Rodriguez & Levy, 2001; Yamazaki & Tanaka, 2005).

The current model also contributes to our understanding of the TD models of dopamine (e.g., Schultz et al., 1997) and classical conditioning (Sutton & Barto, 1990). These models have often given short shrift to issues of stimulus representation, focusing more closely on the properties of the learning algorithm (but see Ludvig et al., 2008). Here, we reveal how the interaction of various stimulus representations in conjunction with the TD learning rule produces a viable model of some of the differences between trace and delay conditioning.

# References

Beylin, A. V., Gandhi, C. C, Wood, G. E., Talk, A. C., Matzel, L. D., & Shors, T. J. (2001). The role of the hippocampus in trace conditioning: Temporal discontinuity or task difficulty? *Neurobiology of Learning & Memory, 76*, 447-61.

Gould, E., Beylin, A., Tanapat, P., Reeves, A., & Shors, T. J. (1999). Learning enhances adult neurogenesis in the hippocampal formation. *Nature Neuroscience, 2*, 260-5.

Grossberg, S., & Schmajuk, N. A. (1989). Neural dynamics of adaptive timing and temporal discrimination during associative learning. *Neural Networks, 2*, 79-102.

Ludvig, E. A., Sutton, R. S., & Kehoe, E. J. (2008). Stimulus representation and the timing of reward-prediction errors in models of the dopamine system. *Neural Computation, 20*, 3034-54.

Machado, A. (1997). Learning the temporal dynamics of behavior. *Psychological Review, 104*, 241-265.

McEchron, M. D., Bouwmeester, H., Tseng, W., Weiss, C., & Disterhoft, J. F. (1998). Hippocampectomy disrupts auditory trace fear conditioning and contextual fear conditioning in the rat. *Hippocampus, 8*, 638-46.

McEchron, M. D., Disterhoft, J. F. (1997). Sequence of single neuron changes in CA1 hippocampus of rabbits during acquisition of trace eyeblink conditioned responses. *Journal of Neurophysiology, 78*, 1030-44.

Moyer, J. R., Jr., Deyo, R. A., & Disterhoft, J. F. (1990). Hippocampectomy disrupts trace eye-blink conditioning in rabbits. *Behavioral Neuroscience, 104*, 243-52.

Pavlov, I. P. (1927). *Conditioned Reflexes*. London: Oxford University Press.

Port, R. L., Romano, A. G., Steinmetz, J. E., Mikhail, A. A., & Patterson, M. M. (1986). Retention and acquisition of classical trace conditioned responses by rabbits with hippocampal lesions. *Behavioral Neuroscience, 100*, 745-752.

Rodriguez, P., & Levy, W. B. (2001). A model of hippocampal activity in trace conditioning: Where's the trace? *Behavioral Neuroscience, 115*, 1224-1238.

Schmajuk, N. A., & DiCarlo, J. J. (1992). Stimulus configuration, classical conditioning, and hippocampal function. *Psychological Review, 99*, 268-305.

Schneiderman, N., & Gormezano, I. (1964). Conditioning of the nictitating membrane of the rabbit as a function of CS-US interval. *Journal of Comparative and Physiological Psychology, 57*, 188-195.

Schultz, W., Dayan, P., & Montague, P. R. (1997). A neural substrate of prediction and reward. *Science, 275*, 1593-9.

Solomon, P. R., Vander Schaaf, E. R., Thompson, R. F., & Weisz, D. J. (1986). Hippocampus and trace conditioning of the rabbit's classically conditioned nictitating membrane response. *Behavioral Neuroscience, 100*, 729-744.

Sutton, R. S. (1988). Learning to predict by the methods of temporal differences. *Machine Learning*, 3, 9-44.

Sutton, R. S., & Barto, A. G. (1990). Time-derivative models of Pavlovian reinforcement. In M. Gabriel & J. Moore (Eds.), *Learning and Computational Neuroscience: Foundations of Adaptive Networks* (pp. 497-537). Cambridge, MA: MIT Press.

Sutton, R. S., & Barto, A. G. (1998). *Reinforcement Learning: An Introduction*. Cambridge, MA: MIT Press.

Tseng, W., Guan, R., Disterhoft, J. F., & Weiss, C. (2004). Trace eyeblink conditioning is hippocampally dependent in mice. *Hippocampus, 14*, 58-65.

Wallenstein, G., Eichenbaum, H., & Hasselmo, M. (1998). The hippocampus as an associator of discontiguous events. *Trends in Neuroscience, 21*, 317-323.

Yamazaki, T., & Tanaka, S. (2005). A neural network model for trace conditioning. *International Journal of Neural Systems, 15*, 23-30.

